# Boltzmann Chains and Hidden Markov Models

**Lawrence K. Saul and Michael I. Jordan**
lksaul@psyche.mit.edu, jordan@psyche.mit.edu
Center for Biological and Computational Learning
Massachusetts Institute of Technology
79 Amherst Street, E10-243
Cambridge, MA 02139

## Abstract

We propose a statistical mechanical framework for the modeling of discrete time series. Maximum likelihood estimation is done via Boltzmann learning in one-dimensional networks with tied weights. We call these networks Boltzmann chains and show that they contain hidden Markov models (HMMs) as a special case. Our framework also motivates new architectures that address particular shortcomings of HMMs. We look at two such architectures: parallel chains that model feature sets with disparate time scales, and looped networks that model long-term dependencies between hidden states. For these networks, we show how to implement the Boltzmann learning rule exactly, in polynomial time, without resort to simulated or mean-field annealing. The necessary computations are done by exact decimation procedures from statistical mechanics.

## 1 INTRODUCTION AND SUMMARY

Statistical models of discrete time series have a wide range of applications, most notably to problems in speech recognition (Juang & Rabiner, 1991) and molecular biology (Baldi, Chauvin, Hunkapiller, & McClure, 1992). A common problem in these fields is to find a probabilistic model, and a set of model parameters, that

account for sequences of observed data. Hidden Markov models (HMMs) have been particularly successful at modeling discrete time series. One reason for this is the powerful learning rule (Baum, 1972), a special case of the Expectation-Maximization (EM) procedure for maximum likelihood estimation (Dempster, Laird, & Rubin, 1977).

In this work, we develop a statistical mechanical framework for the modeling of discrete time series. The framework enables us to relate HMMs to a large family of exactly solvable models in statistical mechanics. The connection to statistical mechanics was first noticed by Sourlas (1989), who studied spin glass models of error-correcting codes. We view the estimation procedure for HMMs as a special (and particularly tractable) case of the Boltzmann learning rule (Ackley, Hinton, & Sejnowski, 1985; Byrne, 1992).

The rest of this paper is organized as follows. In Section 2, we review the modeling problem for discrete time series and establish the connection between HMMs and Boltzmann machines. In Section 3, we show how to quickly determine whether or not a particular Boltzmann machine is tractable, and if so, how to efficiently compute the correlations in the Boltzmann learning rule. Finally, in Section 4, we look at two architectures that address particular weaknesses of HMMs: the modelling of disparate time scales and long-term dependencies.

## 2 MODELING DISCRETE TIME SERIES

A discrete time series is a sequence of symbols $\{j_\ell\}_{\ell=1}^L$ in which each symbol belongs to a finite countable set, i.e. $j_\ell \in \{1, 2, \ldots, m\}$. Given one long sequence, or perhaps many shorter ones, the modeling task is to characterize the probability distribution from which the time series are generated.

### 2.1 HIDDEN MARKOV MODELS

A first-order Hidden Markov Model (HMM) is characterized by a set of $n$ hidden states, an alphabet of $m$ symbols, a transmission matrix $a_{ii'}$, an emission matrix $b_{ij}$, and a prior distribution $\pi_i$ over the initial hidden state. The sequence of states $\{i_\ell\}_{\ell=1}^L$ and symbols $\{j_\ell\}_{\ell=1}^L$ is modeled to occur with probability

$$P(\{i_\ell, j_\ell\}) = \pi_{i_1} a_{i_1 i_2} a_{i_2 i_3} \ldots a_{i_{L-1} i_L} b_{i_1 j_1} b_{i_2 j_2} \ldots b_{i_L j_L}. \tag{1}$$

The modeling problem is to find the parameter values $(a_{ii'}, b_{ij}, \pi_i)$ that maximize the likelihood of observed sequences of training data. We will elaborate on the learning rule in section 2.3, but first let us make the connection to a well-known family of stochastic neural networks, namely Boltzmann machines.

### 2.2 BOLTZMANN MACHINES

Consider a Boltzmann machine with $m$-state visible units, $n$-state hidden units, tied weights, and the linear architecture shown in Figure 1. This example represents the simplest possible Boltzmann "chain", one that is essentially equivalent to a first-order HMM unfolded in time (MacKay, 1994). The transition weights $A_{ii'}$ connect adjacent hidden units, while the emission weights $B_{ij}$ connect each hidden unit to

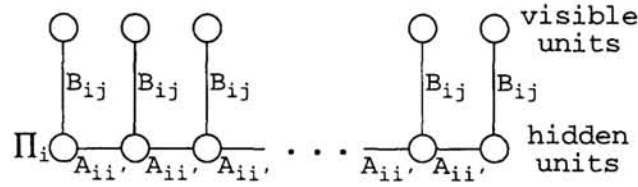

Figure 1: Boltzmann chain with $n$-state hidden units, $m$-state visible units, transition weights $A_{ii'}$, emission weights $B_{ij}$, and boundary weights $\Pi_i$.

its visible counterpart. In addition, boundary weights $\Pi_i$ model an extra bias on the first hidden unit. Each configuration of units represents a state of energy

$$\mathcal{H}[\{i_\ell, j_\ell\}] = -\Pi_{i_1} - \sum_{\ell=1}^{L-1} A_{i_\ell i_{\ell+1}} - \sum_{\ell=1}^{L} B_{i_\ell j_\ell}, \tag{2}$$

where $\{i_\ell\}_{l=1}^{L}$ ($\{j_\ell\}_{l=1}^{L}$) is the sequence of states over the hidden (visible) units. The probability to find the network in a particular configuration is given by

$$P(\{i_\ell, j_\ell\}) = \frac{1}{Z} e^{-\beta \mathcal{H}}, \tag{3}$$

where $\beta = 1/T$ is the inverse temperature, and the partition function

$$Z = \sum_{\{i_\ell, j_\ell\}} e^{-\beta \mathcal{H}} \tag{4}$$

is the sum over states that normalizes the Boltzmann distribution, eq. (3).

Comparing this to the HMM distribution, eq. (1), it is clear that any first-order HMM can be represented by the Boltzmann chain of figure 1, provided we take[1]

$$A_{ii'} = T \ln a_{ii'}, \quad B_{ij} = T \ln b_{ij}, \quad \Pi_i = T \ln \pi_i. \tag{5}$$

Later, in Section 4, we will consider more complicated chains whose architectures address particular shortcomings of HMMs. For now, however, let us continue to develop the example of figure 1, making explicit the connection to HMMs.

## 2.3 LEARNING RULES

In the framework of Boltzmann learning (Williams & Hinton, 1990), the data for our problem consist of sequences of states over the visible units; the goal is to find the weights $(A_{ii'}, B_{ij}, \Pi_i)$ that maximize the likelihood of the observed data. The likelihood of a sequence $\{j_\ell\}$ is given by the ratio

$$P(\{j_\ell\}) = \frac{P(\{i_\ell, j_\ell\})}{P(\{i_\ell\}|\{j_\ell\})} = \frac{e^{-\beta \mathcal{H}}/Z}{e^{-\beta \mathcal{H}}/Z_c} = \frac{Z_c}{Z}, \tag{6}$$

where $Z_c$ is the clamped partition function

$$Z_c = \sum_{\{i_\ell\}} e^{-\beta \mathcal{H}}. \tag{7}$$

Note that the sum in $Z_c$ is only over the hidden states in the network, while the visible states are clamped to the observed values $\{j_\ell\}$.

The Boltzmann learning rule adjusts the weights of the network by gradient-ascent on the log-likelihood. For the example of figure 1, this leads to weight updates

$$\Delta A_{ii'} = \eta\beta \sum_{\ell=1}^{L-1} \left[ \langle \delta_{ii_\ell}\delta_{i'i_{\ell+1}} \rangle_c - \langle \delta_{ii_\ell}\delta_{i'i_{\ell+1}} \rangle \right], \tag{8}$$

$$\Delta B_{ij} = \eta\beta \sum_{\ell=1}^{L} \left[ \langle \delta_{ii_\ell}\delta_{jj_\ell} \rangle_c - \langle \delta_{ii_\ell}\delta_{jj_\ell} \rangle \right], \tag{9}$$

$$\Delta \Pi_i = \eta\beta \left[ \langle \delta_{ii_1} \rangle_c - \langle \delta_{ii_1} \rangle \right], \tag{10}$$

where $\delta_{ij}$ stands for the Kronecker delta function, $\eta$ is a learning rate, and $\langle \cdot \rangle$ and $\langle \cdot \rangle_c$ denote expectations over the free and clamped Boltzmann distributions.

The Boltzmann learning rule may also be derived as an Expectation–Maximization (EM) algorithm. The EM procedure is an alternating two-step method for maximum likelihood estimation in probability models with hidden and observed variables. For Boltzmann machines in general, neither the E-step nor the M-step can be done exactly; one must estimate the necessary statistics by Monte Carlo simulation (Ackley et al., 1985) or mean-field theory (Peterson & Anderson, 1987). In certain special cases (e.g. trees and chains), however, the necessary statistics can be computed to perform an exact E-step (as shown below). While the $M$-step in these Boltzmann machines cannot be done exactly, the weight updates can be approximated by gradient descent. This leads to learning rules in the form of eqs. (8–10).

HMMs may be viewed as a special case of Boltzmann chains for which both the $E$-step *and* the $M$-step are analytically tractable. In this case, the maximization in the $M$-step is performed subject to the constraints $\sum_i e^{\beta\Pi_i} = 1$, $\sum_{i'} e^{\beta A_{ii'}} = 1$, and $\sum_j e^{\beta B_{ij}} = 1$. These constraints imply $Z = 1$ and lead to closed-form equations for the weight updates in HMMs.

## 3   EXACT METHODS FOR BOLTZMANN LEARNING

The key technique to compute partition functions and correlations in Boltzmann chains is known as decimation. The idea behind decimation[2] is the following. Consider three units connected in series, as shown in Figure 2a. Though not directly connected, the end units have an effective interaction that is mediated by the middle one. In fact, the two weights in series exert the same influence as a single *effective* weight, given by

$$e^{\beta A_{ii''}} = \sum_{i'} e^{\beta A^{(1)}_{ii'} + \beta A^{(2)}_{i'i''} + \beta B_{i'}}. \tag{11}$$

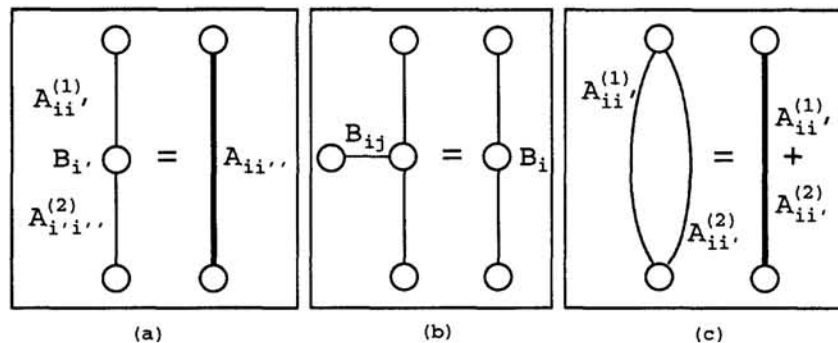

Figure 2: Decimation, pruning, and joining in Boltzmann machines.

Replacing the weights in this way amounts to integrating out, or decimating, the degree of freedom represented by the middle unit. An analogous rule may be derived for the situation shown in Figure 2b. Summing over the degrees of freedom of the dangling unit generates an effective bias on its parent, given by

$$e^{\beta B_i} = \sum_j e^{\beta B_{ij}}. \tag{12}$$

We call this the pruning rule. Another type of equivalence is shown in Figure 2c. The two weights in parallel have the same effect as the sum total weight

$$A_{ii'} = A_{ii'}^{(1)} + A_{ii'}^{(2)}. \tag{13}$$

We call this the joining rule. It holds trivially for biases as well as weights.

The rules for decimating, pruning, and joining have simple analogs in other types of networks (e.g. the law for combining resistors in electric circuits), and the strategy for exploiting them is a familiar one. Starting with a complicated network, we iterate the rules until we have a simple network whose properties are easily computed. A network is tractable for Boltzmann learning if it can be reduced to any pair of connected units. In this case, we may use the rules to compute all the correlations required for Boltzmann learning. Clearly, the rules do not make all networks tractable; certain networks (e.g. trees and chains), however, lend themselves naturally to these types of operations.

## 4   DESIGNER NETS

The rules in section 3 can be used to quickly assess whether or not a network is tractable for Boltzmann learning. Conversely, they can be used to design networks that are computationally tractable. This section looks at two networks designed to address particular shortcomings of HMMs.

### 4.1   PARALLEL CHAINS AND DISPARATE TIME SCALES

An important problem in speech recognition (Juang et al., 1991) is how to "combine feature sets with fundamentally different time scales." Spectral parameters, such

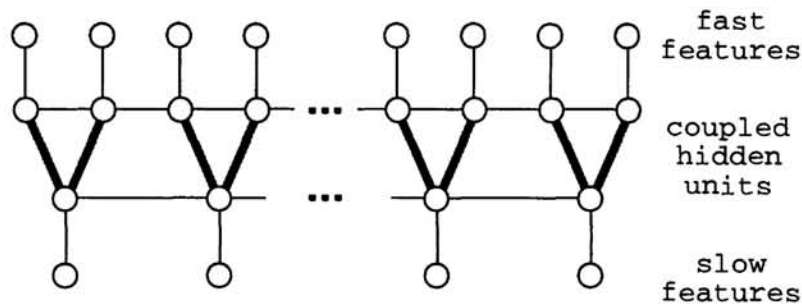

Figure 3: Coupled parallel chains for features with different time scales.

as the cepstrum and delta-cepstrum, vary on a time scale of 10 msec; on the other hand, prosodic parameters, such as the signal energy and pitch, vary on a time scale of 100 msec. A model that takes into account this disparity should avoid two things. The first is redundancy—in particular, the rather lame solution of oversampling the nonspectral features. The second is overfitting. How might this arise? Suppose we have trained two separate HMMs on sequences of spectral and prosodic features, knowing that the different features "may not warrant a single, unified Markov chain" (Juang et al., 1991). To exploit the correlation between feature sets, we must now couple the two HMMs. A naive solution is to form the Cartesian product of their hidden state spaces and resume training. Unfortunately, this results in an explosion in the number of parameters that must be fit from the training data. The likely consequences are overfitting and poor generalization.

Figure 3 shows a network for modeling feature sets with disparate time scales—in this case, a 2:1 disparity. Two parallel Boltzmann chains are coupled by weights that connect their hidden units. Like the transition and emission weights within each chain, the coupling weights are tied across the length of the network. Note that coupling the time scales in this way introduces far fewer parameters than forming the Cartesian product of the hidden state spaces. Moreover, the network is tractable by the rules of section 3. Suppose, for example, that we wish to compute the correlation between two neighboring hidden units in the middle of the network. This is done by first pruning all the visible units, then repeatedly decimating hidden units from both ends of the network.

Figure 4 shows typical results on a simple benchmark problem, with data generated by an artificially constructed HMM. We tested the parallel chains model on 10 training sets, with varying levels of built-in correlation between features. A two-step method was used to train the parallel chains. First, we set the coupling weights to zero and trained each chain by a separate Baum-Welch procedure. Then, after learning in this phase was complete, we lifted the zero constraints and resumed training with the full Boltzmann learning rule. The percent gain in this second phase was directly related to the degree of correlation built into the training data, suggesting that the coupling weights were indeed capturing the correlation between feature sets. We also compared the performance of this Boltzmann machine versus that of a simple Cartesian-product HMM trained by an additional Baum-Welch procedure. While in both cases the second phase of learning led to reduced training error, the Cartesian product HMMs were decidedly more prone to overfitting.

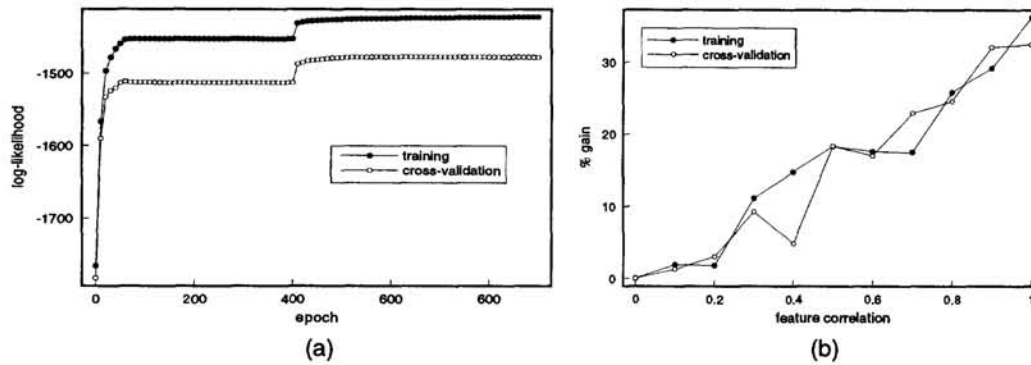

Figure 4: (a) Log-likelihood versus epoch for parallel chains with 4-state hidden units, 6-state visible units, and 100 hidden–visible unit pairs (per chain). The second jump in log-likelihood occurred at the onset of Boltzmann learning (see text). (b) Percent gain in log-likelihood versus built-in correlation between feature sets.

## 4.2  LOOPS AND LONG-TERM DEPENDENCIES

Another shortcoming of first-order HMMs is that they cannot exhibit long-term dependencies between the hidden states (Juang et al., 1991). Higher-order and duration-based HMMs have been used in this regard with varying degrees of success. The rules of section 3 suggest another approach—namely, designing tractable networks with limited long-range connectivity. As an example, Figure 5a shows a Boltzmann chain with an internal loop and a long-range connection between the first and last hidden units. These extra features could be used to enforce known periodicities in the time series. Though tractable for Boltzmann learning, the loops in this network do not fit naturally into the framework of HMMs. Figure 5b shows learning curves for a toy problem, with data generated by another looped network.

Carefully chosen loops and long-range connections provide additional flexibility in the design of probabilistic models for time series. Can networks with these extra features capture the long-term dependencies exhibited by real data? This remains an important issue for future research.

### Acknowledgements

We thank G. Hinton, D. MacKay, P. Stolorz, and C. Williams for useful discussions. This work was funded by ATR Human Information Processing Laboratories, Siemens Corporate Research, and NSF grant CDA-9404932.

## Footnotes

[1]Note, however, that the reverse statement—that for any set of parameters, this Boltzmann chain can be represented as an HMM—is *not* true. The weights in the Boltzmann chain represent arbitrary energies between $\pm\infty$, whereas the HMM parameters represent probabilities that are constrained to obey sum rules, such as $\sum_{i'} a_{ii'} = 1$. The Boltzmann chain of figure 1 therefore has slightly more degrees of freedom than a first-order HMM. An interpretation of these extra degrees of freedom is given by MacKay (1994).

[2] A related method, the transfer matrix, is described by Stolorz (1994).

### References

D. H. Ackley, G. E. Hinton, and T. J. Sejnowski. (1985) A Learning Algorithm for Boltzmann Machines. *Cog. Sci.* **9**: 147–160.

P. Baldi, Y. Chauvin, T. Hunkapiller, and M. A. McClure. (1992) *Proc. Nat. Acad. Sci. (USA)* **91**: 1059-1063.

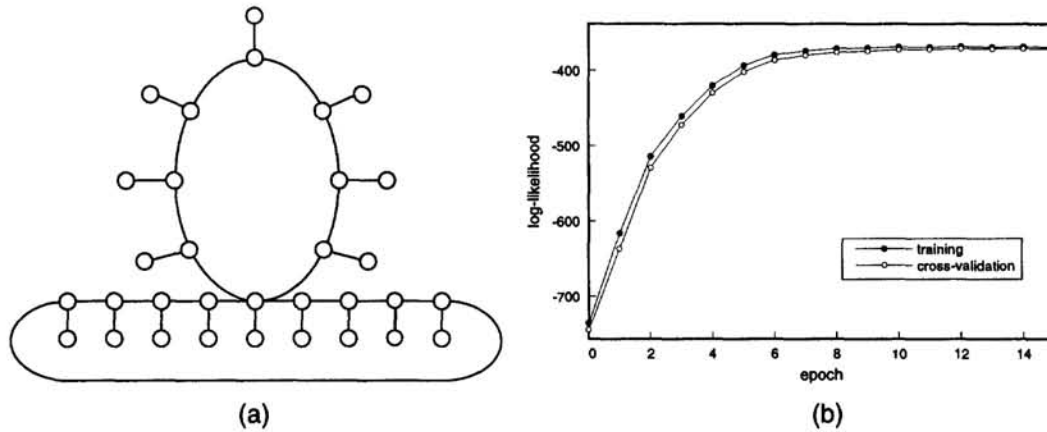

Figure 5: (a) Looped network. (b) Log-likelihood versus epoch for a looped network with 4-state hidden units, 6-state visible units, and 100 hidden–visible unit pairs.

L. Baum. (1972) An Inequality and Associated Maximization Technique in Statistical Estimation of Probabilistic Functions of Markov Processes, *Inequalities* **3**:1–8.

Byrne, W. (1992) Alternating Minimization and Boltzmann Machine Learning. *IEEE Trans. Neural Networks* **3**:612–620.

A. P. Dempster, N. M. Laird, and D. B. Rubin. (1977) Maximum Likelihood from Incomplete Data via the EM Algorithm. *J. Roy. Statist. Soc. B*, **39**:1–38.

C. Itzykson and J. Drouffe. (1991) *Statistical Field Theory*, Cambridge: Cambridge University Press.

B. H. Juang and L. R. Rabiner. (1991) Hidden Markov Models for Speech Recognition, *Technometrics* **33**: 251–272.

D. J. MacKay. (1994) Equivalence of Boltzmann Chains and Hidden Markov Models, submitted to *Neural Comp.*

C. Peterson and J. R. Anderson. (1987) A Mean Field Theory Learning Algorithm for Neural Networks, *Complex Systems* **1**:995–1019.

L. Saul and M. Jordan. (1994) Learning in Boltzmann Trees. *Neural Comp.* **6**: 1174–1184.

N. Sourlas. (1989) Spin Glass Models as Error Correcting Codes. *Nature* **339**: 693–695.

P. Stolorz. (1994) Links Between Dynamic Programming and Statistical Physics for Heterogeneous Systems, JPL/Caltech preprint.

C. Williams and G. E. Hinton. (1990) Mean Field Networks That Learn To Discriminate Temporally Distorted Strings. *Proc. Connectionist Models Summer School*: 18–22.